# Unbiased Estimator of Shape Parameter for Spiking Irregularities under Changing Environments

**Keiji Miura**
Kyoto University
JST PRESTO

**Masato Okada**
University of Tokyo
JST PRESTO
RIKEN BSI

**Shun-ichi Amari**
RIKEN BSI

## Abstract

We considered a gamma distribution of interspike intervals as a statistical model for neuronal spike generation. The model parameters consist of a time-dependent firing rate and a shape parameter that characterizes spiking irregularities of individual neurons. Because the environment changes with time, observed data are generated from the time-dependent firing rate, which is an unknown function. A statistical model with an unknown function is called a semiparametric model, which is one of the unsolved problem in statistics and is generally very difficult to solve. We used a novel method of estimating functions in information geometry to estimate the shape parameter without estimating the unknown function. We analytically obtained an optimal estimating function for the shape parameter independent of the functional form of the firing rate. This estimation is efficient without Fisher information loss and better than maximum likelihood estimation.

## 1 Introduction

The firing patterns of cortical neurons look very noisy [1]. Consequently, probabilistic models are necessary to describe these patterns [2, 3, 4]. For example, Baker and Lemon showed that the firing patterns recorded from motor areas can be explained using a continuous-time rate-modulated gamma process [5]. Their model had a rate parameter, $\xi$, and a shape parameter, $\kappa$, that was related to spiking irregularity. $\xi$ was assumed to be a function of time because it depended largely on the behavior of the monkey. $\kappa$ was assumed to be unique to individual neurons and constant over time.

The assumption that $\kappa$ is unique to individual neurons is also supported by other studies [6, 7, 8]. However, these indirect supports are not conclusive. Therefore, we need to accurately estimate $\kappa$ to make the assumption more reliable. If the assumption is correct, neurons may be identified by $\kappa$ estimated from the spiking patterns, and $\kappa$ may provide useful information about the function of a neuron. In other words, it may be possible to classify neurons according to functional firing patterns rather than static anatomical properties. Thus, it is very important to accurately estimate $\kappa$ in the field of neuroscience.

In reality, however, it is very difficult to estimate all the parameters in the model from

the observed spike data. The reason for this is that the unknown function for the time-dependent firing rate, $\xi(t)$, has infinite degrees of freedom. This kind of estimation problem is called the semiparametric model [9] and is one of the unsolved problems in statistics. Are there any ingenious methods of estimating $\kappa$ accurately to overcome this difficulty?

Ikeda pointed out that the problem we need to consider is the semiparametric model [10]. However, the problem remains unsolved. There is a method called estimating functions [11, 12] for semiparametric problems, and a general theory has been developed [13, 14, 15] from the viewpoint of information geometry [16, 17, 18]. However, the method of estimating functions cannot be applied to our problem in its original form.

In this paper, we consider the semiparametric model suggested by Ikeda instead of the continuous-time rate-modulated gamma process. In this discrete-time rate-modulated model, the firing rate varies for each interspike interval. This model is a mixture model and can represent various types of interspike interval distributions by adjusting its weight function. The model can be analyzed by using the method of estimating functions for semiparametric models.

Various attempts have been made to solve semiparametric models. Neyman and Scott pointed out that the maximum likelihood method does not generally provide a consistent estimator when the number of parameters and observations are the same [19]. In fact, we show that maximum likelihood estimation for our problem is biased. Ritov and Bickel considered asymptotic attainability of information bound purely mathematically [20, 21]. However, their results were not practical for application to our problem. Amari and Kawanabe showed a practical method of estimating finite parameters of interest without estimating an unknown function [15]. This is the method of estimating functions. If this method can be applied, $\kappa$ can be estimated consistently independent of the functional form of a firing rate.

In this paper, we show that the model we consider here is the "exponential form" defined by Amari and Kawanabe [15]. However, an asymptotically unbiased estimating function does not exist unless multiple observations are given for each firing rate, $\xi$. We show that if multiple observations are given, the method of estimating functions can be applied. In that case, the estimating function of $\kappa$ can be analytically obtained, and $\kappa$ can be estimated consistently independent of the functional form of a firing rate. In general, estimation using estimating functions is not efficient. However, for our problem, this method yielded an optimal estimator in the sense of Fisher information [15]. That is, we obtained an efficient estimator.

## 2  Simple case

We considered the following statistical model of inter spike intervals proposed by Ikeda [10]. Interspike intervals are generated by a gamma distribution whose mean firing rate changes over time. The mean firing rate $\xi$ at each observation is determined randomly according to an unknown probability distribution, $k(\xi)$. The model is described as

$$p(T;\kappa,k(\xi)) = \int q(T;\xi,\kappa)k(\xi)d\xi, \tag{1}$$

where

$$
\begin{aligned}
q(T;\xi,\kappa) &= \frac{(\xi\kappa)^\kappa}{\Gamma(\kappa)}T^{\kappa-1}e^{-\xi\kappa T} \\
&= e^{\xi(-\kappa T)+(\kappa-1)\log(T)-(-\kappa\log(\xi\kappa)+\log\Gamma(\kappa))} \\
&\equiv e^{\xi s(T,\kappa)+r(T,\kappa)-\psi(\kappa,\xi)}.
\end{aligned}
\tag{2}
$$

Here, $T$ denotes an interspike interval. We defined $s$, $r$, and $\psi$ as

$$s(T, \kappa) = -\kappa T, \tag{3}$$
$$r(T, \kappa) = (\kappa - 1)\log(T), \text{ and} \tag{4}$$
$$\psi(\kappa, \xi) = -\kappa \log(\xi \kappa) + \log \Gamma(\kappa) \tag{5}$$

to demonstrate that the model is the exponential form defined by Amari and Kawanabe [15]. Note that this type of model is called a semiparametric model because it has both unknown finite parameters, $\kappa$, and function, $k(\xi)$.

In this mixture model, $\{\xi^{(1)}, \xi^{(2)}, \ldots\}$ is an unknown sequence where $\xi$ is independently and identically distributed according to a probability density function $k(\xi)$. Then, $l$-th observation $T^{(l)}$ is distributed according to $q(T^{(l)}; \xi^{(l)}, \kappa)$. In effect, $T$ is independently and identically distributed according to $p(T; \kappa, k(\xi))$.

An estimating function is a function of $\kappa$ whose zero-crossing provides an estimate of $\kappa$, analogous to the derivative with respect to $\kappa$ of the log-likelihood function. Note that the zero-crossings of the derivatives of the log-likelihood function with respect to parameters provide an maximum likelihood estimator.

Let us calculate the estimating function following Amari and Kawanabe [15] to estimate $\kappa$ without estimating $k(\xi)$. They showed that for the exponential form of mixture distributions, the estimating function, $u^I$, is given by the projection of the score function, $u = \partial_\kappa \log p$, as

$$
\begin{aligned}
u^I(T, \kappa) &= u - E[u|s] \\
&= (\partial_\kappa s - E[\partial_\kappa s|s]) \cdot E_\xi[\xi|s] + \partial_\kappa r - E[\partial_\kappa r|s] \\
&= \partial_\kappa r - E[\partial_\kappa r|s],
\end{aligned}
\tag{6}
$$

where

$$E_\xi[\xi|s] = \frac{\int \xi k(\xi) \exp(\xi \cdot s - \psi) d\xi}{\int k(\xi) \exp(\xi \cdot s - \psi) d\xi}. \tag{7}$$

The relation,

$$E[\partial_\kappa s|s] = \frac{s}{\kappa} = -T = \partial_\kappa s, \tag{8}$$

holds because the number of random variables, $T$, and $s$ are the same. For the same reason,

$$E[\partial_\kappa r|s] = \log(T) = \partial_\kappa r. \tag{9}$$

Then,

$$u^I = 0. \tag{10}$$

This means that the set of estimating functions is an empty set. Therefore, we proved that no asymptotically unbiased estimating function of $\kappa$ exists for the model.

Two or more random variables may be needed. Let us consider the multivariate model described as

$$p(T_1, ..., T_n; \kappa, k(\xi_1, ..., \xi_n)) = \int \prod_{i=1}^n q(T_i; \xi_i, \kappa) k(\xi_1, ..., \xi_n) d\xi. \tag{11}$$

Here, the number of random variables and s are also the same, and $u^I$ becomes an empty set.

This result can be understood intuitively as follows. When the mean, $\mu$, and variance, $\sigma$, of a normal distribution are estimated from a single observation, $x$, they are estimated as $\mu = x$ and $\sigma = 0$. Similarly, $\xi$ and $\kappa$ of a gamma distribution, $q(T; \xi, \kappa)$, are estimated from a single observation, $T$, as $\xi = \frac{1}{T}$ and $\kappa = \infty$ corresponding to 0 variance. Two or more observations are required to estimate $\kappa$. For the semiparametric model considered in this section, only one observation is given for each $\xi$. Two or more observations are needed for each $\xi$.

## 3 Cases with multiple observations for each $\xi$

Next we consider the case where $m$ observations are given for each $\xi^{(l)}$, which may be distributed according to $k(\xi)$. Here, a consistent estimator of $\kappa$ exists. Let $\{T\} = \{T_1, \ldots, T_m\}$ be the $m$ observations, which are generated from the same distribution specified by $\xi$ and $\kappa$. We have $N$ such observations $\{T^{(l)}\}, l = 1, \ldots, N$, with a common $\kappa$ and different $\xi^{(l)}$. Thus, $\{T_1^{(l)}, \ldots, T_m^{(l)}\}$ are generated from the same firing rate $\xi^{(l)}$. Let us take one $\{T\}$. The probability model can be written as

$$p(\{T\}; \kappa, k(\xi)) = \int \prod_{i=1}^{m} q(T_i; \xi, \kappa) k(\xi) d\xi, \tag{12}$$

where

$$\begin{aligned}
\prod_{i=1}^{m} q(T_i; \xi, \kappa) &= \prod_{i=1}^{m} \frac{(\xi\kappa)^{\kappa}}{\Gamma(\kappa)} T_i^{\kappa-1} e^{-\xi\kappa T_i} \\
&= e^{\xi(-\kappa \sum_{i=1}^{m} T_i) + (\kappa-1) \sum_{i=1}^{m} \log(T_i) - (-m\kappa \log(\xi\kappa) + m \log \Gamma(\kappa))} \\
&\equiv e^{(\xi \cdot s(\{T\}, \kappa) + r(\{T\}, \kappa) - \psi(\kappa, \xi))}.
\end{aligned} \tag{13}$$

We defined $s$, $r$, and $\psi$ as

$$s(\{T\}, \kappa) = -\kappa \sum_{i=1}^{m} T_i, \tag{14}$$

$$r(\{T\}, \kappa) = (\kappa-1) \sum_{i=1}^{m} \log(T_i), \text{ and} \tag{15}$$

$$\psi(\kappa, \xi) = -m\kappa \log(\xi\kappa) + m \log \Gamma(\kappa). \tag{16}$$

Then, the estimating function is given by

$$\begin{aligned}
u^I(\{T\}, \kappa) &= u - E[u|s] \\
&= (\partial_\kappa s - E[\partial_\kappa s|s]) \cdot E_\xi[\xi|s] + \partial_\kappa r - E[\partial_\kappa r|s] \\
&= \partial_\kappa r - E[\partial_\kappa r|s] \\
&= \sum_{i=1}^{m} \log(T_i) - m E[\log(T_1)|s],
\end{aligned} \tag{17}$$

where we used

$$E[\partial_\kappa s|s] = \frac{s}{\kappa} = \partial_\kappa s. \tag{18}$$

To calculate the conditional expectation of $\log T_1$, let us use Bayes's Theorem:

$$p(T|s) = \frac{p(T, s)}{p(s)}. \tag{19}$$

By transforming random variables, $(T_1, T_2, T_3, ..., T_m)$, into $(s, T_2, T_3, ..., T_m)$, we have

$$\begin{aligned}
p(s) &= \int \prod_i q(T_i; \xi, \kappa) \delta(s + \kappa \sum_{i=1}^{m} T_i) k(\xi) d\xi dT \\
&= \prod_{i=1}^{m-1} B(i\kappa, \kappa) \frac{(-s)^{m\kappa-1}}{\Gamma(\kappa)^m} \int \xi^{m\kappa} e^{s\xi} k(\xi) d\xi.
\end{aligned} \tag{20}$$

where the beta function is defined as

$$B(x,y) = \frac{\Gamma(x)\Gamma(y)}{\Gamma(x+y)} = \frac{(x-1)!(y-1)!}{(x+y-1)!}. \tag{21}$$

Similarly, we have

$$
\begin{aligned}
E[\log(T_1)|s] &= \int \log(T_1) \prod_{i=1}^{m} q(T_i)\delta(s + \kappa \sum_{i=1}^{m} T_i)k(\xi)d\xi dT \frac{1}{p(s)} \\
&= \log(-\frac{s}{\kappa}) - \phi(m\kappa) + \phi(\kappa), \tag{22}
\end{aligned}
$$

where the digamma function is defined as

$$\phi(\kappa) = \frac{\Gamma'(\kappa)}{\Gamma(\kappa)}. \tag{23}$$

Note that $E[\log(T_1)|s]$ does not depend on the unknown function, $k(\xi)$. Thus, we have

$$u^I(\{T\},\kappa) = \sum_{i=1}^{m} \log(T_i) - m\log(\sum_{i=1}^{m} T_i) + m\phi(m\kappa) - m\phi(\kappa). \tag{24}$$

The form of $u^I$ can be understood as follows. If we scale $T$ as $t = \xi T$, we have $E[t] = 1$. Then, we can show that $u^I$ does not depend on $\xi$, because

$$\log(T) - E[\log T|s] = \log(t) - E[\log t|s]. \tag{25}$$

This implies that we can estimate $\kappa$ without estimating $\xi$. The method of estimating function only works for gamma distributions. It crucially depends on the fact that the estimating function is invariant under scaling of $T$.

$\kappa$ can be estimated consistently from $N$ independent observations, $\{T^{(l)}\} = \{T_1^{(l)}, \ldots, T_m^{(l)}\}, l = 1, \ldots, N$, as the value of $\kappa$ that solves

$$\sum_{l=1}^{N} u^I(\{T^{(l)}\}, \hat{\kappa}) = 0. \tag{26}$$

In fact, the expectation of $u^I$ is 0 independent of $k(\xi)$:

$$
\begin{aligned}
E[u^I] &= \int (\int \prod_{i=1}^{m} q(T_i; \xi, \kappa) u^I dT) k(\xi)d\xi \\
&= \int E_q[u^I|s]p(s)ds \cdot \int k(\xi)d\xi \\
&= \int E_q[\log t - E[\log t|s]|s]p(s)ds \\
&= 0, \tag{27}
\end{aligned}
$$

where $E_q$ denotes the expectation for $\prod_{i=1}^{m} q(t_i; 1, \kappa)$.

$u^I$ yields an efficient estimating function [15, 21]. An efficient estimator is one whose variance attains the Cramer-Rao lower bound asymptotically. Thus, there is no estimator of $\kappa$ whose mean-square estimation error is smaller than that given by $u^I$. As $u^I$ does not depend on $k(\xi)$, it is the optimal estimating function whatever $k(\xi)$ is, or whatever the sequence $\xi^{(1)}, \ldots, \xi^{(N)}$ is.

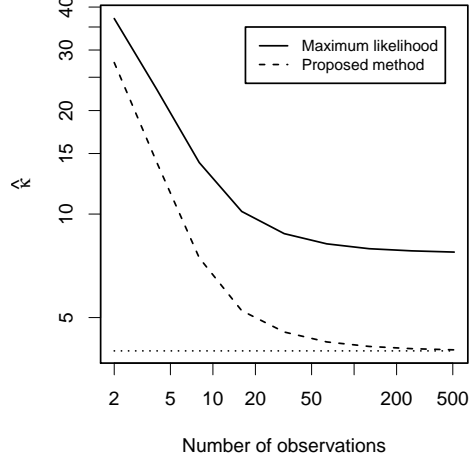

Figure 1: Biases of $\hat{\kappa}$ for maximum likelihood estimation and proposed method for $m = 2$. The dotted line represents the true value, $\kappa = 4$. The maximum likelihood estimation is biased even when an infinite number of observations are given while the estimating function is asymptotically unbiased.

The maximum likelihood estimation for this problem is given by

$$u^{MLE} = \sum_{i=1}^{m} \log(T_i) + m \log(\hat{\xi}) + m \log \kappa - m\phi(\kappa), \tag{28}$$

where

$$\frac{1}{\hat{\xi}} = \frac{1}{m} \sum_{i=1}^{m} T_i. \tag{29}$$

$u^{MLE}$ is similar to $u^I$ but different in terms of constant. As a result, the maximum likelihood estimator $\hat{\kappa}$ is biased (Figure 1).

So far, we have assumed that the firing rates for $m$ observations are the same. Instead, let us consider a case where the firing rates have some relation. For example, consider the case where $E_q[t_1] = 2E_q[t_2]$. The model can be written as

$$p(t_1, t_2; \kappa, k(\xi)) = \int q(t_1; \xi, \kappa)q(t_2; 2\xi, \kappa)k(\xi)d\xi. \tag{30}$$

This model can be derived from Eq. (12) by rescaling as $T_1 = t_1$ and $T_2 = 2t_2$. Note that $q(2T; \xi, \kappa) = q(T; 2\xi, \kappa)$ because $T$ always appears as $\xi T$ in $q(T; \xi, \kappa)$. Thus, Eq. (12) includes various kinds of models.

## 4 General case

Let us consider a general case where the firing rate changes stepwise. That is, $\{\xi_1, \ldots, \xi_n\}$ is distributed according to $k(\{\xi\}) = k(\xi_1, \ldots, \xi_n)$ and $m_a$ observations are given for each $\xi_a$. The model can be written as

$$p(\{T\}; \kappa, k(\{\xi\}))$$
$$= \int \prod_{i_1=1}^{m_1} q(T_{i_1}^{(1)}; \xi_1, \kappa) \prod_{i_2=1}^{m_2} q(T_{i_2}^{(2)}; \xi_2, \kappa) \ldots \prod_{i_n=1}^{m_n} q(T_{i_n}^{(n)}; \xi_n, \kappa)k(\{\xi\})d\xi_1 d\xi_2 \ldots d\xi_n,$$

where

$$\prod_{i_1=1}^{m_1} q(T_{i_1}^{(1)};\xi_1,\kappa) \prod_{i_2=1}^{m_2} q(T_{i_2}^{(2)};\xi_2,\kappa) \ldots \prod_{i_n=1}^{m_n} q(T_{i_n}^{(n)};\xi_n,\kappa)$$

$$= \exp(\xi_1(-\kappa \sum_{i_1=1}^{m_1} T_{i_1}^{(1)}) + \xi_2(-\kappa \sum_{i_2=1}^{m_2} T_{i_2}^{(2)}) + \ldots + \xi_n(-\kappa \sum_{i_n=1}^{m_n} T_{i_n}^{(n)})$$

$$+ (\kappa-1)(\sum_{i_1=1}^{m_1} \log T_{i_1}^{(1)} + \sum_{i_2=1}^{m_2} \log T_{i_2}^{(2)} + \ldots + \sum_{i_n=1}^{m_n} \log T_{i_n}^{(n)})$$

$$+ \sum_{a=1}^{n} m_a \kappa \log(\xi_a) + \sum_{a=1}^{n} m_a \kappa \log(\kappa) - \sum_{a=1}^{n} m_a \log \Gamma(\kappa)). \tag{32}$$

We defined $s_a$, $r$, and $\psi$ as

$$s_a(\{T^{(a)}\},\kappa) = -\kappa \sum_{i_a=1}^{m_a} T_{i_a}^{(a)}, \tag{33}$$

$$r(\{T\},\kappa) = (\kappa-1) \sum_{a=1}^{n} (\sum_{i_a=1}^{m_a} \log T_{i_a}^{(a)}), \tag{34}$$

$$\psi(\kappa,\{\xi\}) = -\sum_{a=1}^{n} m_a \kappa \log(\xi_a) - \sum_{a=1}^{n} m_a \kappa \log(\kappa) + \sum_{a=1}^{n} m_a \log \Gamma(\kappa). \tag{35}$$

Then,

$$u^I(\{T\},\kappa) = u - E[u|s]$$
$$= (\partial_\kappa s - E[\partial_\kappa s|s]) \cdot E[\xi|s] + \partial_\kappa r - E[\partial_\kappa r|s]$$
$$= \partial_\kappa r - E[\partial_\kappa r|s] \tag{36}$$
$$= \sum_{a=1}^{n} \{\sum_{i_a=1}^{m_a} \log T_{i_a}^{(a)} - m_a \log(\sum_{i_a=1}^{m_a} T_{i_a}^{(a)}) + m_a \phi(m_a\kappa) - m_a \phi(\kappa)\}.$$

Thus, $\kappa$ is estimated with equal weight for every observation. Note that the conditional expectations can be calculated independently for each set of random variables. $u^I$ yields an efficient estimating function. As this does not depend on $k(\{\xi\})$, $u^I$ is the optimal estimating function at any $k(\{\xi\})$. There is no information loss. Note that $k(\{\xi\})$ can include correlations among $\xi_a$'s. Nevertheless, the result is very similar to that of the previous section.

## 5 Summary and discussion

We estimated the shape parameter, $\kappa$, of the semiparametric model suggested by Ikeda without estimating the firing rate, $\xi$. The maximum likelihood estimator is not consistent for this problem because the number of nuisance parameters, $\xi$, increases with increasing observations, $T$. We showed that Ikeda's model is the exponential form defined by Amari and Kawanabe [15] and can be analyzed by a method of estimating functions for semiparametric models. We found that an estimating function does not exist unless multiple observations are given for each firing rate, $\xi$. If multiple observations are given, a method of estimating functions can be applied. In that case, the estimating function of $\kappa$ can be analytically obtained, and $\kappa$ can be estimated consistently independent of the functional form of the firing rate, $k(\xi)$. In general, the estimating function is not efficient. However, this method provided an optimal estimator in the sense of Fisher information for our problem. That is, we obtained an efficient estimator.

**Acknowledgments**

We are grateful to K. Ikeda for his helpful discussions. This work was supported in part by grants from the Japan Society for the Promotion of Science (Nos. 14084212 and 16500093).

# References

[1] G. R. Holt, W. R. Softky, C. Koch, and R. J. Douglas, Comparison of discharge variability in vitro and in vivo in cat visual cortex neurons, J. Neurophysiol., Vol. 75, pp. 1806-14, 1996.

[2] H. C. Tuckwell, Introduction to theoretical neurobiology: volume 2, nonlinear and stochastic theories, Cambridge University Press, Cambridge, 1988.

[3] Y. Sakai, S. Funahashi, and S. Shinomoto, Temporally correlated inputs to leaky integrate-and-fire models can reproduce spiking statistics of cortical neurons, Neural Netw., Vol. 12, pp. 1181-1190, 1999.

[4] D. R. Cox and P. A. W. Lewis, The statistical analysis of series of events, Methuen, London, 1966.

[5] S. N. Baker and R. N. Lemon, Precise spatiotemporal repeating patterns in monkey primary and supplementary motor areas occur at chance levels, J. Neurophysiol., Vol. 84, pp. 1770-80, 2000.

[6] S. Shinomoto, K. Shima, and J. Tanji, Differences in spiking patterns among cortical neurons, Neural Comput.,Vol. 15, pp. 2823-42, 2003.

[7] S. Shinomoto, Y. Miyazaki, H. Tamura, and I. Fujita, Regional and laminar differences in in vivo firing patterns of primate cortical neurons, J. Neurophysiol., in press.

[8] S. Shinomoto, K. Miura, and S. Koyama, A measure of local variation of inter-spike intervals, Biosystems, Vol. 79, pp. 67-72, 2005.

[9] J. Pfanzagl, Estimation in semiparametric models, Springer-Verlag, Berlin, 1990.

[10] K. Ikeda, Information geometry of interspike intervals in spiking neurons, Neural Comput., in press.

[11] V. P. Godambe, An optimum property of regular maximum likelihood estimation, Ann. Math. Statist., Vol. 31, pp. 1208-1211, 1960.

[12] V. P. Godambe (ed.), Estimating functions, Oxford University Press, New York, 1991.

[13] S. Amari, Dual connections on the Hilbert bundles of statistical models, In C. T. J. Dodson (ed.), Geometrization of statistical theory, pp. 123-152, University of Lancaster Department of Mathematics, Lancaster, 1987.

[14] S. Amari and M. Kumon, Estimation in the presence of infinitely many nuisance parameters - geometry of estimating functions, Ann. Statist., Vol. 16, pp. 1044-1068, 1988.

[15] S. Amari and M. Kawanabe, Information geometry of estimating functions in semi-parametric statistical models, Bernoulli, Vol. 3, pp. 29-54, 1997.

[16] H. Nagaoka and S. Amari, Differential geometry of smooth families of probability distributions, Technical Report 82-7, University of Tokyo, 1982.

[17] S. Amari and H. Nagaoka, Methods of information geometry, American Mathematical Society, Providence, RI, 2001.

[18] S. Amari, Information geometry on hierarchy of probability distributions, IEEE Transactions on Information Theory, Vol. 47, pp. 1701-1711, 2001.

[19] J. Neyman and E. L. Scott, Consistent estimates based on partially consistent observations, Econometrica, Vol. 32, pp. 1-32, 1948.

[20] Y. Ritov and P. J. Bickel, Achieving information bounds in non and semiparametric models, Ann. Statist., Vol. 18, pp. 925-938, 1990.

[21] P. J. Bickel, C. A. J. Klaassen, Y. Ritov, and J. A. Wellner, Efficient and adaptive estimation for semiparametric models, Johns Hopkins University Press, Baltimore, MD, 1993.
